# Grammar Transfer in a Second Order Recurrent Neural Network

**Michiro Negishi**
Department of Psychology
Rutgers University
101 Warren St. Smith Hall #301
Newark, NJ 07102
*negishi@psychology.rutgers.edu*

**Stephen José Hanson**
Psychology Department
Rutgers University
101 Warren St. Smith Hall #301
Newark, NJ 07102
*jose@psychology.rutgers.edu*

## Abstract

It has been known that people, after being exposed to sentences generated by an artificial grammar, acquire implicit grammatical knowledge and are able to transfer the knowledge to inputs that are generated by a modified grammar. We show that a second order recurrent neural network is able to transfer grammatical knowledge from one language (generated by a Finite State Machine) to another language which differ *both* in vocabularies and syntax. Representation of the grammatical knowledge in the network is analyzed using linear discriminant analysis.

## 1 Introduction

In the field of artificial grammar learning, people are known to be able to transfer grammatical knowledge to a new language which consists of a new vocabulary [6]. Furthermore, this effect persists even when the new strings violate the syntactic rule slightly as long as they are similar to the old strings [1]. It has been shown in the past studies that recurrent neural networks also have the ability to generalize previously acquired knowledge to novel inputs. For instance, Dienes *et al.* ([2]) showed that a neural network can generalize abstract knowledge acquired in one domain to a new domain. They trained the network to predict the next input symbol in grammatical sequences in the first domain, and showed that the network was able to learn to predict grammatical sequences in the second domain more effectively than it would have learned them without the prior learning. During the training in the second domain, they had to freeze the weights of a part of the network to prevent catastrophic forgetting. They used this simulation paradigm to emulate and analyze domain transfer, effect of similarity between training and test sequences, and the effect of n-gram information in human data. Hanson *et al.* ([5]) also showed that a prior learning of a grammar facilitates the learning of a new grammar in the cases where either the syntax or the vocabulary was kept constant.

In this study we investigate grammar transfer by a neural network, where both syntax and vocabularies are different from the source grammar to the target grammar. Unlike Dienes *et al.*'s network, all weights in the network are allowed to change dur-

ing the learning of the target grammar, which allows us to investigate interference as well as transfer from the source grammar to the target grammar.

## 2   Simulation Design

### 2.1   The Grammar Transfer Task

In the following simulations, a neural network is trained with sentences that are generated by a Finite State Machine (FSM) and is tested whether the learning of sentences generated by another FSM is facilitated. Four pairs of FSMs used for the grammar transfer task are shown in Fig. 2. In each FSM diagram, symbols (*e.g.* A, B, C, ...) denote words, numbers represent states, a state number with an incoming arrow with no state numbers at the arrow foot (*e.g.* state 1 in the left FSM in Fig. 2A) signifies the initial state, and numbers in circles (*e.g.* state 3 in the left FSM in Fig. 2A) signify the accepting states. In each pair of diagrams, transfer was tested in both directions: from the left FSM to the right FSM, and to the opposite direction. Words in a sentence are generated by an FSM and presented to the network one word at a time. At each time, the next word is selected randomly from next possible words (or end of sentence where possible) at the current FSM state with the equal probability, and the FSM state is updated to the next state. The sentence length is limited to 20 words, excluding START.

The task for the network is to predict the correct termination of sentences. If the network is to predict that the sentence ends with the current input, the activity of the output node of the network has to be above a threshold value, otherwise the output has to be below another threshold value. Note that if a FSM is at an accepting state but can further transit to another state, the sentence may or may not end. Therefore, the prediction may succeed or fail. However, the network will eventually learn to yield higher values when the FSM is at an accepting state than when it is not. After the network learns each training sentence, it is tested with randomly generated 1000 sentences and the training session is completed only when the network makes correct end point judgments for all sentences. Then the network is trained with sentences generated by another FSM. The extent of transfer is measured by the reduction of the number of sentences required to train the network on an FSM after a prior learning of another FSM, compared to the number of sentences required to train the network on the current FSM from scratch.

### 2.2   The Network Architecture and the Learning Algorithm

The network is a second order recurrent neural network, with an added hidden layer that receives first order connections from the input layer (Fig. 1). The network has an input layer with seven nodes (A, B, C, ... F, and START), an output layer with one node, an input hidden layer with four nodes, a state hidden layer with four nodes, and a feedback layer with four nodes. Recurrent neural networks are often used for modeling syntactic processing [3]. Second order networks are suited for processing languages generated by FSMs [4]. Learning is carried out by the weight update rule for recurrent networks developed by Williams and Zipser ([7]), extended to second order connections ([4]) where necessary. The learning rate and the momentum are 0.2 and 0.8, respectively. High and low thresholds are initialized to 0.20 and 0.17 respectively and are adapted after the network have processed the test sentences as follows. The high threshold is modified to the minimum value yielded for all end points in the test sentences minus a margin (0.01). The low threshold is modified to the high threshold minus another margin (0.02). These thresholds are used in the next training and test.

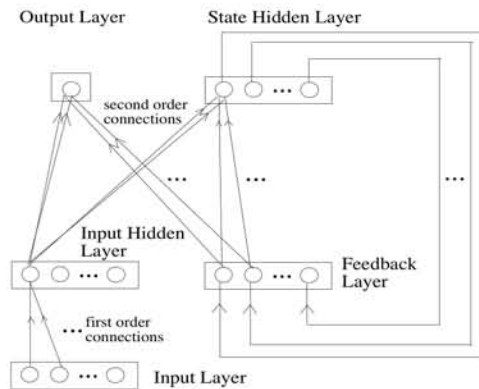

Figure 1: A second order recurrent network used in simulations. The network consists of an input layer that receives words, an output layer that predicts sentence ends, two hidden layers (an input hidden layer and a state hidden layer), and a feedback layer that receives a copy of the state hidden layer activities.

# 3 The Simulation Results

## 3.1 The Transfer Effects

Numbers of required trainings and changes in number of trainings averaged over 20 networks with different initial weights are shown in Fig. 2. Numbers in parentheses are standard errors of number of trainings. Changes are shown with either a "+" sign (increase) or a "-" sign (reduction). For instance, Fig. 2A shows that it required 14559 sentence presentations for the network to learn the left FSM after the network was trained on the right FSM. On the other hand, it required 20995 sentence presentation for the network to learn the left FSM from the scratch. Therefore there was 30.7% reduction in the transfer direction from right to left. Note that the network was trained only once on sentences from the source grammar to the criteria and then only once on the sentences from the target grammar. Thus after the completion of the target grammar learning, the knowledge about the source grammar is disrupted to some extent. To show that the network eventually learns both grammars, number of required training was examined for more than one cycle. After ten cycles, number of required trainings was reduced to 0.13% (not shown).

## 3.2 Representation of Grammatical Knowledge

To analyze the representation of grammatical knowledge in the network, Linear Discriminant Analysis (LDA) was applied to hidden layer activities. LDA is a technique which finds sets of coefficients that defines a linear combination of input variables that can be used to discriminate among sets of input data that belong to different categories. Linear combinations of hidden layer node activities using these coefficients provide low-dimensional views of hidden layer activities that best separate specified categories (*e.g.* grammatical functions). In this respect, LDA is similar to Principal Component Analysis (PCA) except that PCA finds dimensions along which the data have large variances, whereas LDA finds dimensions which differentiate the specified categories.

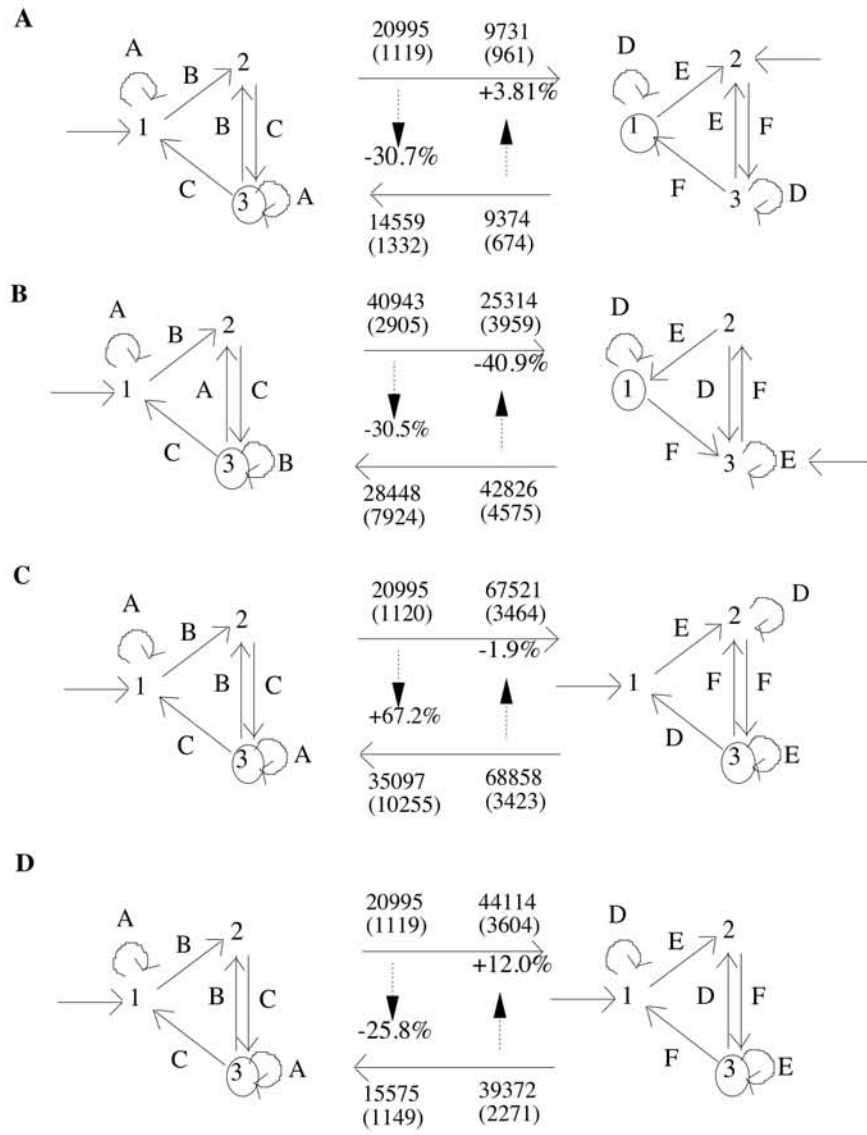

Figure 2: Initial savings observed in various grammar transfer tasks. Numbers are required number of training averaged over 20 networks with different initial weights. Numbers in parentheses are standard errors. Numbers shown with "%" are change in number of training due to transfer. A negative change means reduction (positive transfer) and a positive change means increase (negative transfer, or interference).

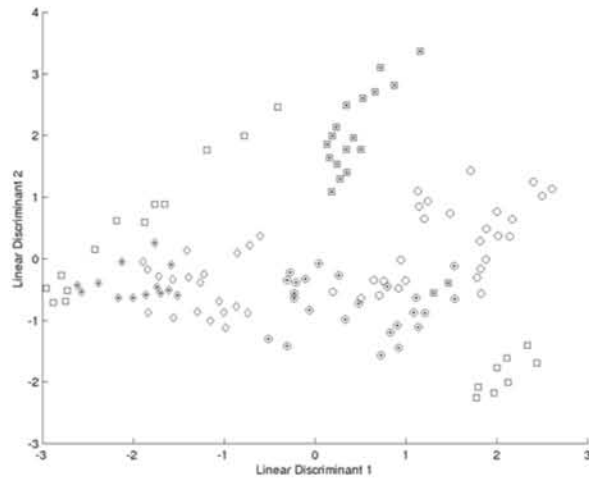

Figure 3: State space organization for a grammar transfer task (a case of Fig. 2B). State space activities corresponding to FSM states 1, 2, and 3 are plotted with squares, diamonds, and circles, respectively. State space activities that belong to the target FSM have dots in the plots, whereas those that belong to the source FSM do not have fill in patterns.

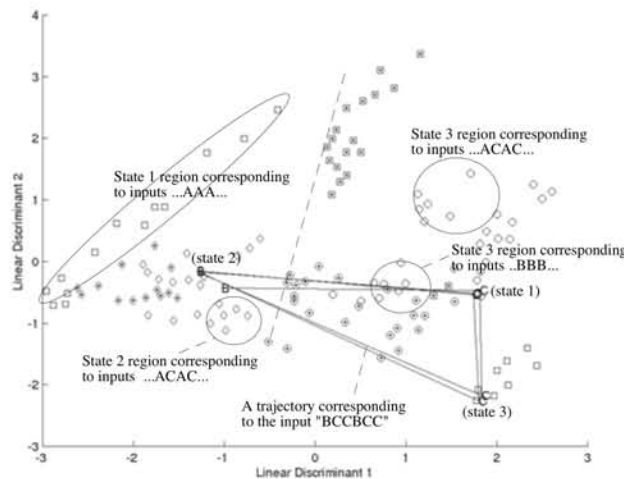

Figure 4: Trajectories corresponding to loops in Fig. 2B in the state hidden layer state space. The broken line corresponds to a hypothetical shared discrimination cue a hypothetical boundary described in 4. It is the between white diamonds and white circles (*i.e.* states 2 and 3 in the source grammar), as well as it can be one of the discrimination boundaries between diamonds with dots and squares with dots (*i.e.* states 2 and 1 in the target grammar). The triangular shape shows the three FSM state trajectory corresponding to inputs BCCBCC.... Ellipses show to state space activities involved in one state loops (at state 1 and at state 3) and two state loops (at state 2 and 3).

# 4   Discussion

In the first grammar transfer task (Fig. 2A), only the initial and the accepting states in the FSMs were different, so the frequency distribution of subsequences of words were very similar except for short sentences. In this case, 31% saving was observed in one transfer direction although there was little change in required training in the other direction. In the second grammar transfer task, directions of all arcs in the FSMs were reversed. Therefore the mirror images of sentences accepted in one grammar were accepted in the other grammar. Although the grammars were very different, there were significant amount of overlaps in the permissible short subsequences. In this case, there were 31% and 41% savings in training. In the third and fourth grammar transfer tasks, the source and the target grammars shared less subsequences. In the third case (Fig. 2C) for instance, the subsequences were very different because the source grammar had two one-state loops (at states 1 and 3) with the same word A, whereas two one-state loops in the target grammar consisted of different words (D and E). In this case, there was little change in the number of learnings required in one transfer direction but there was 67% increase in the other direction. In the fourth case (Fig 2. D), there was 26% reduction in one direction but there was 12% increase in the other direction in the number of learnings required. From these observations we hypothesize that, as in the case of syntax transfer ([5]), if the acquired grammar allows frequent subsequence of words that appears in the target grammar (after the equivalent symbol sets are substituted) the transfer is easier and thus there are more savings.

What is the source of savings in grammar transfer? It is tempting to say that, as in the vocabulary transfer task ([5]), the source of savings is the organization of the state hidden layer activity which directly reflects the FSM states. Fig. 3 shows the state space organization after the grammar transfer shown in Fig. 2B. Fig. 4 shows the change in the state hidden layer activities drawn over the state space organization. The triangular lines are the trajectories as the network receives BCCBCC, which creates the 3-state loops (231)(231) in the FSM. Regions of trajectories corresponding to the 2-state loop (23) and two 1-state loops (1) and (3) are also shown in Fig. 4, although the trajectory lines are not shown to avoid a cluttered figure. It can be seen that state space activities that belong to different FSM state loops tend to be distinct even when they belong to the same FSM state, although there seem to be some tendencies that they are allocated in vicinities. Unlike in the vocabulary transfer, regions belonging to different FSM loops tend to be interspersed by regions that belong to the other grammar, causing state space structure to be more fragmented. Furthermore, we found that there was no significant correlation between the correct rate of the linear discrimination with respect to FSM states (which reflects the extent to which the state space organization reflects the FSM states) and savings (not shown).

One could reasonably argue that the saving is not due to transfer of grammatical knowledge but is due to some more low-level processing specific to neural networks. For instance, the network may have to move weight values to an appropriate range at the first stage of the source grammar learning, which might become unnecessary for the leaning of the target grammar. We conducted a simulation to examine the effect of altering the initial random weights using the source and target grammars. The space limitation does not permit us to present the details, but we did not observe the effect of initializing the bias and the weights to appropriate ranges.

If neither the state space organization nor the lower-level statistics was not the source of savings, what was transferred? As already mentioned, state space organization observed in grammar transfer task is more fragmented than that observed

in vocabulary transfer task (Fig. 3). These fragmented regions have to be discriminated as far as each region (which represents a combination of the current network state and the current vocabulary) has to yield a different network state. State hidden nodes provide clues for the discrimination by placing boundaries in the network state space. Boundary lines collectively define regions in the state space which correspond to sets of state-vocabulary combinations that should be treated equivalently in terms of the given task. These boundaries can be shared: for instance, a hypothetical boundary shown by a broken line in the Fig. 4 can be the discrimination boundary between white diamonds and white circles (*i.e.* states 2 and 3 in the source grammar), as well as it can be one of the discrimination boundaries between diamonds with dots and squares with dots (*i.e.* states 2 and 1 in the target grammar). We speculate that shared boundaries may be the source of savings. That is, boundaries created for the source grammar learning can be used, possibly with some modifications, as one of the boundaries for the target grammar. In other words, the source of savings may not be as high level as FSM state space but some lower level features at the syntactic processing level.

## 5 Conclusion

We investigated the ability of a recurrent neural network to transfer grammatical knowledge of a previously acquired language to another. We found that the network was able to transfer the grammatical knowledge to a new grammar with a slightly different syntax defined over a new vocabulary (grammar transfer). The extent of transfer seemed to depend on the subsequences of symbols generated by the two grammars, after the equivalence sets are translated, although the results presented in this paper are admittedly very restricted in the type of syntax covered and the size of syntactic rules and vocabularies. We hypothesize that the ability of the network to transfer grammatical knowledge comes from sharing discrimination boundaries of input and vocabulary combinations. In sum, we hope to have demonstrated that neural networks do not simply learn associations among input symbols but they acquire structural knowledge from inputs.

**References**

[1] Brooks, L. R., and Vokey, J. R. (1991) Abstract analogies and abstracted grammars: Comments on Reber (1989) and Mathews *et al.* (1090). *Journal of Experimental Psychology: General*, **120**, 316-323.

[2] Dienes, Z., Altmann, and G., Gao, S-J. (1999) Mapping across domains without feedback: A neural network model of transfer of implicit knowledge, *Cognitive Science* **23**, 53-82.

[3] Elman, J. L. (1991) Distributed representation, simple recurrent neural networks, and grammatical structure. *Machine Learning*, **7**, 195-225.

[4] Giles, C. L., Miller, C. B., Chen, D., Chen, H. H., Sun, G. Z., and Lee, Y. C. (1992) Learning and Extracting Finite State Automata with Second-Order Recurrent Neural Networks, it Neural Computation, 4, 393-495.

[5] Hanson, S. J., Negishi, M., (2001) The emergence of explicit knowledge (symbols & rules) in (associationist) neural networks, Submitted.

[6] Reber, A. (1969) Transfer of syntactic structure in synthetic languages. *Journal of Experimental Psychology*, **81**, 115-119.

[7] Williams, R. J. and Zipser, D. (1989) A learning algorithm for continually running fully recurrent neural networks, *Neural Computation*, **1** (2), 270.
